# Learning Influence among Interacting Markov Chains

**Dong Zhang**
IDIAP Research Institute
CH-1920 Martigny, Switzerland
zhang@idiap.ch

**Daniel Gatica-Perez**
IDIAP Research Institute
CH-1920 Martigny, Switzerland
gatica@idiap.ch

**Samy Bengio**
IDIAP Research Institute
CH-1920 Martigny, Switzerland
bengio@idiap.ch

**Deb Roy**
Massachusetts Institute of Technology
Cambridge, MA 02142, USA
dkroy@media.mit.edu

## Abstract

We present a model that learns the influence of interacting Markov chains within a team. The proposed model is a dynamic Bayesian network (DBN) with a two-level structure: individual-level and group-level. Individual level models actions of each player, and the group-level models actions of the team as a whole. Experiments on synthetic multi-player games and a multi-party meeting corpus show the effectiveness of the proposed model.

## 1   Introduction

In multi-agent systems, individuals within a group coordinate and interact to achieve a goal. For instance, consider a basketball game where a team of players with different roles, such as attack and defense, collaborate and interact to win the game. Each player performs a set of individual actions, evolving based on their own dynamics. A group of players interact to form a team. Actions of the team and its players are strongly correlated, and different players have different influence on the team. Taking another example, in conversational settings, some people seem particularly capable of driving the conversation and dominating its outcome. These people, skilled at establishing the leadership, have the largest influence on the group decisions, and often shift the focus of the meeting when they speak [8].

In this paper, we quantitatively investigate the influence of individual players on their team using a dynamic Bayesian network, that we call two-level influence model. The proposed model explicitly learns the influence of individual player on the team with a two-level structure. In the first level, we model actions of individual players. In the second one, we model team actions as a whole. The model is then applied to determine (a) the influence of players in multi-player games, and (b) the influence of participants in meetings.

The paper is organized as follows. Section 2 introduces the two-level influence model. Section 3 reviews related models. Section 4 presents results on multi-player games, and Section 5 presents results on a meeting corpus. Section 6 provides concluding remarks.

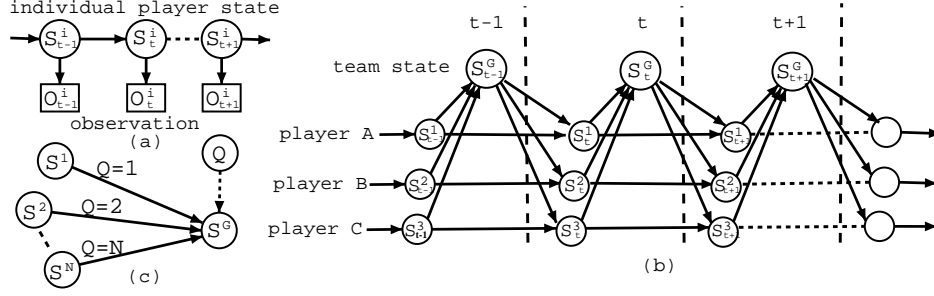

Figure 1: (a) Markov Model for individual player. (b) Two-level influence model (for simplicity, we omit the observation variables of individual Markov chains, and the switching parent variable $Q$). (c) Switching parents. $Q$ is called a switching parent of $S^G$, and $\{S^1 \cdots S^N\}$ are conditional parents of $S^G$. When $Q = i$, $S^i$ is the only parent of $S^G$.

## 2   Two-level Influence Model

The proposed model, called two-level influence model, is a dynamic Bayesian network (DBN) with a two-level structure: the *player* level and the *team* level (Fig. 1). The player level represents the actions of individual players, evolving based on their own Markovian dynamics (Fig. 1 (a)). The team level represents group-level actions (the action belongs to the team as a whole, not to a particular player). In Fig. 1 (b), the arrows up (from players to team) represent the influence of the individual actions on the group actions, and the arrows down (from team to players) represent the influence of the group actions on the individual actions. Let $O^i$ and $S^i$ denote the observation and state of the $i^{th}$ player respectively, and $S^G$ denotes the team state. For $N$ players, and observation sequences of identical length $T$, the joint distribution of our model is given by

$$P(S,O) = \prod_{i=1}^{N} P(S_1^i) \prod_{t=1}^{T} \prod_{i=1}^{N} P(O_t^i|S_t^i) \prod_{t=1}^{T} P(S_t^G|S_t^1 \cdots S_t^N) \prod_{t=2}^{T} \prod_{i=1}^{N} P(S_t^i|S_{t-1}^i, S_{t-1}^G). \quad (1)$$

Regarding the player level, we model the actions of each individual with a first-order Markov model (Fig. 1 (a)) with one observation variable $O^i$ and one state variable $S^i$. Furthermore, to capture the dynamics of all the players interacting as a team, we add a hidden variable $S^G$ (team state), which is responsible to model the group-level actions. Different from individual player state that has its own Markovian dynamics, team state is not directly influenced by its previous state . $S^G$ could be seen as the aggregate behaviors of the individuals, yet provides a useful level of description beyond individual actions. There are two kinds of relationships between the team and players: (1) The team state at time $t$ influences the players' states at the next time (down arrow in Fig. 1 (b)). In other words, the state of the $i^{th}$ player at time $t + 1$ depends on its previous state as well as on the team state, i.e., $P(S_{t+1}^i|S_t^i, S_t^G)$. (2) The team state at time $t$ is influenced by all the players' states at the current time (up arrow in Fig. 1 (b)), resulting in a conditional state transition distribution $P(S_t^G|S_t^1 \cdots S_t^N)$.

To reduce the model complexity, we add one hidden variable $Q$ in the model, to switch parents for $S^G$. The idea of switching parent (also called Bayesian multi-nets in [3]) is as follows: a variable -$S^G$ in this case- has a set of parents $\{Q, S^1 \cdots S^N\}$ (Fig. 1(c)). $Q$ is the switching parent that determines which of the other parents to use, conditioned on the current value of the switching parent. $\{S^1 \cdots S^N\}$ are the conditional parents. In Fig. 1(c), $Q$ switches the parents of $S^G$ among $\{S^1 \cdots S^N\}$, corresponding to the distribution

$$P(S_t^G|S_t^1 \cdots S_t^N) = \sum_{i=1}^{N} P(S_t^G, Q = i|S_t^1 \cdots S_t^N) \quad (2)$$

$$= \sum_{i=1}^{N} P(Q=i|S_t^1 \cdots S_t^N) P(S_t^G|S_t^i \cdots S_t^N, Q=i) \quad (3)$$

$$= \sum_{i=1}^{N} P(Q=i) P(S_t^G|S_t^i) = \sum_{i=1}^{N} \alpha_i P(S_t^G|S_t^i). \quad (4)$$

From Eq. 3 to Eq. 4, we made two assumptions: (i) $Q$ is independent of $\{S^1 \cdots S^N\}$; and (ii) when $Q = i$, $S_t^G$ only depends on $S_t^i$. The distribution over the switching-parent variable $P(Q)$ essentially describes how much influence or contribution the state transitions of the player variables have on the state transitions of the team variable. We refer to $\alpha_i = P(Q = i)$ as the influence value of the $i^{th}$ player. Obviously, $\sum_{i=1}^{N} \alpha_i = 1$. If we further assume that all player variables have the same number of states $N_S$, and the team variable has $N_G$ possible states, the joint log probability is given by

$$\log P(S,O) = \underbrace{\sum_{i=1}^{N} \sum_{j=1}^{N_S} z_{j,1}^i \cdot \log P(S_1^i = j)}_{initial\ probability} + \underbrace{\sum_{t=1}^{T} \sum_{i=1}^{N} \sum_{j=1}^{N_S} z_{j,t}^i \cdot \log P(O_t^i|S_t^i = j)}_{emission\ probability}$$

$$+ \underbrace{\sum_{t=2}^{T} \sum_{i=1}^{N} \sum_{j=1}^{N_S} \sum_{k=1}^{N_S} \sum_{g=1}^{N_G} z_{j,t}^i \cdot z_{k,t-1}^i \cdot z_{g,t-1}^G \cdot \log P(S_t^i = j|S_{t-1}^i = k, S_{t-1}^G = g)}_{group\ influence\ on\ individual\ transition}$$

$$+ \underbrace{\sum_{t=1}^{T} \sum_{k=1}^{N_S} \sum_{g=1}^{N_G} z_{g,t}^G \cdot z_{k,t}^i \cdot \log\{\sum_{i=1}^{N} \alpha_i P(S_t^G = g|S_t^i = k)\}}_{individual\ influence\ on\ group}, \quad (5)$$

where the indicator variable $z_{j,t} = 1$ if $S_t = j$, otherwise $z_{j,t} = 0$. We can see that the model has complexity $O(T \cdot N \cdot N_G \cdot N_S^2)$. For $T = 2000, N_S = 10, N_G = 5, N = 4$, a total of $10^6$ operations is required, which is still tractable.

For the model implementation, we used the Graphical Models Toolkit (GMTK) [4], a DBN system for speech, language, and time series data. Specifically, we used the switching parents feature of GMTK, which greatly facilitates the implementation of the two-level model to learn the influence values using the Expectation Maximization (EM) algorithm. Since EM has the problem of local maxima, good initialization is very important. To initialize the emission probability distribution in Eq. 5, we first train individual action models (Fig. 1 (a)) by pooling all observation sequences together. Then we use the trained emission distribution from the individual action model to initialize the emission distribution of the two-level influence model.This procedure is beneficial because we use data from all individual streams together, and thus have a larger amount of training data for learning.

## 3 Related Models

The proposed two-level influence model is related to a number of models, namely mixed-memory Markov model (MMM) [14, 11], coupled HMM (CHMM) [13], influence model [1, 2, 6] and dynamical systems trees (DSTs) [10]. MMMs decompose a complex model into mixtures of simpler ones, for example, a K-order Markov model, into mixtures of first-order models: $P(S_t|S_{t-1} S_{t-2} \cdots S_{t-K}) = \sum_{i=1}^{K} \alpha_i P(S_t|S_{t-i})$. The CHMM models interactions of multiple Markov chains by directly linking the current state of one stream with the previous states of all the streams (including itself): $P(S_t^i|S_{t-1}^1 S_{t-1}^2 \cdots S_{t-1}^N)$. However, the model becomes computationally intractable for more than two streams. The influence model [1, 2, 6] simplifies the state transition distribution of the CHMM into a

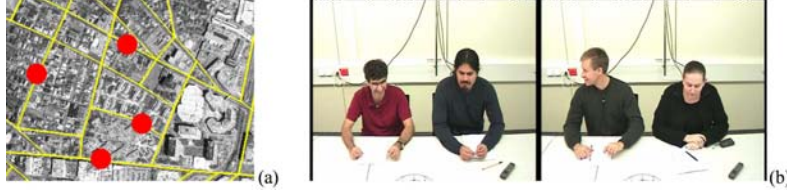

Figure 2: (a) A snapshot of the multi-player games: four players move along the pathes labeled in the map. (b) A snapshot of four-participant meetings.

convex combination of pairwise conditional distributions, i.e., $P(S_t^i|S_{t-1}^1 S_{t-1}^2 \cdots S_{t-1}^N) = \sum_{j=1}^N \alpha_{ji} P(S_t^i|S_{t-1}^j)$. We can see that influence model and MMM take the same strategy to reduce complex models with large state spaces to a combination of simpler ones with smaller state spaces. In [2, 6], the influence model was used to analyze speaking patterns in conversations (i.e., turn-taking) to determine how much influence one participant has on others. In such model, $\alpha_{ji}$ is regarded as the influence of the $j^{th}$ player on the $i^{th}$ player.

All these models, however, limit themselves to modeling the interactions between individual players, *i.e.,* the influence of *one player on another player*. The proposed two-level influence model extends these models by using the group-level variable $S^G$ that allows to model the influence between *all the players and the team*: $P(S_t^G|S_t^1 S_t^2 \cdots S_t^N) = \sum_{i=1}^N \alpha_i P(S_t^G|S_t^i)$, and additionally conditioning the dynamics of each player on the team state: $P(S_{t+1}^i|S_t^i, S_t^G)$.

DSTs [10] have a tree structure that models interacting processes through the parent hidden Markov chains. There are two differences between DSTs and our model: (1) In DSTs, the parent chain has its own Markovian dynamics, while the team state of our model is not directly influenced by the previous team state. Thus, our model captures the emergent phenomena in which the group action is "nothing more" than the aggregate behaviors of individuals, yet it provides a useful level of representation beyond individual actions. (2) The influence between players and team in our model is "bi-direction" (up and down arrows in Fig. 1(b)). In DSTs, the influence between child and parent chains is "uni-direction": parent chains could influence child chains, while child chains could not influence their parent chains.

## 4  Experiments on Synthetic Data

We first test our model on multi-player synthetic games, in which four players (labeled A-D) move along a number of predetermined paths manually labeled in a map (Fig. 2(a)), based on the following rules:

- Game I: Player $A$ moves randomly. Player $B$ and $C$ are meticulously following player $A$. Player $D$ moves randomly.
- Game II: Player $A$ moves randomly. Player $B$ is meticulously following player $A$. Player $C$ moves randomly. Player $D$ is meticulously following player $C$.
- Game III: All four players, $A$, $B$, $C$ and $D$, move randomly.

A follower moves randomly until it lies on the same path of its target, and after that it tries to reach the target by following the target's direction. The initial positions and speeds of players are randomly generated. The observation of an individual player is its motion trajectory in the form of a sequence of positions, $(x_1, y_1), (x_2, y_2) \cdots (x_t, y_t)$, each of which belongs to one of 20 predetermined paths in the map. Therefore, we set $N_S = 20$. The number of team states is set to $N_G = 5$. In experiments, we found that the final results were not sensitive to the specific number of team states for this dataset in a wide range. The length of each game sequence is $T = 2000$ frames. EM iterations were stopped once

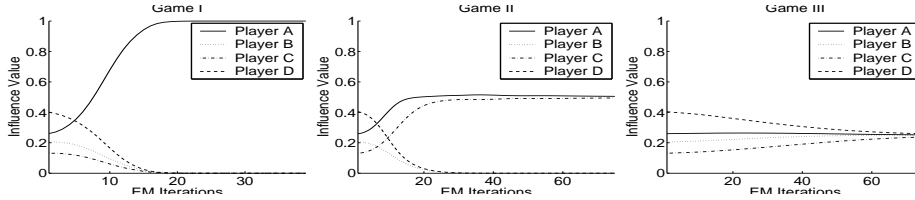

Figure 3: Influence values with respect to the EM iterations in different games.

the relative difference in the global log likelihood was less than 2%.

Fig. 3 shows the learned influence value for each of the four players in the different games with respect to the number of EM iterations. We can see that for Game I, player $A$ is the leader player based on the defined rules. The final learned influence value for player $A$ is almost 1, while the influence for the rest three players are almost 0. For Game II, player $A$ and player $C$ are both leaders based on the defined rules. The learned influence values for player $A$ and $C$ are indeed close to $0.5$, which indicates they have similar influence on the team. For Game III, the four players are moving randomly, and the learned influence values are around $0.25$, which indicates that all players have similar influence on the team. The results on these toy data suggest that our model is capable of learning sensible values for $\{\alpha_i\}$, in good agreement with the concept of influence we have described before.

## 5   Experiments on Meeting Data

As an application of the two-level influence model, we investigate the influence of participants in meetings. Status, dominance, and influence are important concepts in social psychology for which our model could be particularly suitable in a (dynamic) conversational setting [8]. We used a public meeting corpus (available at http://mmm.idiap.ch), which consists of 30 five-minute four-participant meetings collected in a room equipped with synchronized multi-channel audio and video recorders [12]. A snapshot of the meeting is shown in Fig. 2 (b). These meetings have pre-defined topics and an action agenda, designed to ensure discussions and monologues. Manual speech transcripts are also available. We first describe how we manually collected influence judgements, and the performance measure we used. We then report our results using audio and language features, compared with simple baseline methods.

### 5.1   Manually Labeling Influence Values and the Performance Measure

The manual annotation of influence of meeting participants is to some degree a subjective task, as a definite ground-truth does not exist. In our case, each meeting was labeled by three independent annotators who had no access to any information about the participants (e.g. job titles and names). This was enforced to avoid any bias based on prior knowledge of the meeting participants (e.g. a student would probably assign a large influence value to his supervisor). After watching an entire meeting, the three annotators were asked to assign a probability-based value (ranging from 0 to 1, all adding up to 1) to meeting participants, which indicated their influence in the meeting (Fig. 5(b-d)). From the three annotations, we computed the pairwise Kappa statistics [7], a commonly used measure for inter-rate agreement. The obtained pairwise Kappa ranges between $0.68$ and $0.72$, which demonstrates a good agreement among the different annotators. We estimated the ground-truth influence values by averaging the results from the three annotators (Fig. 5(a)).

We use Kullback-Leibler (KL) divergence to evaluate the results. For the $j^{th}$ meeting, given an automatically determined influence distribution $\tilde{P}(Q)$, and the ground truth influence distribution $P(Q)$, the KL divergence is given by: $D^j(\tilde{P}\|P) =$

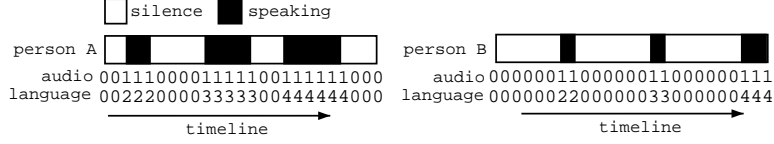

Figure 4: Illustration of state sequences using audio and language features respectively: Using audio, there are two states: speaking and silence. Using language, the number of states equals PLSA topics plus one silence state.

$\sum_{i=1}^{N} \tilde{P}(Q = i) \log_2 \frac{\tilde{P}(Q=i)}{P(Q=i)}$, where $N$ is the number of participants. The smaller $D^j$, the better the performance (if $\tilde{P} = P \Rightarrow D^j = 0$). Note that KL divergence is not symmetric. We calculate the average KL divergence for all the meetings: $D = \frac{1}{M} \sum_{j=1}^{M} D^j(\tilde{P} \| P)$, where $M$ is the number of meetings.

## 5.2 Audio and Language Features

We first extract audio features useful to detect speaking turns in conversations. We compute the SRP-PHAT measure using the signals from a 8-microphone array [12], which is a continuous value indicating the speech activity from a particular participant. We use a Gaussian emission probability, and set $N_S = 2$, each state corresponding to speaking and non-speaking (silence), respectively (Fig. 4).

Additionally, language features were extracted from manual transcripts. After removing stop words, the meeting corpus contains 2175 unique terms. We then employed *probabilistic latent semantic analysis* (PLSA) [9], which is a language model that projects documents in the high-dimensional bag-of-words space into a topic-based space of lower dimension. Each dimension in this new space represents a "topic", and each document is represented as a mixture of topics. In our case, a document corresponds to one speech utterance $(t_s, t_e, w_1 w_2 \cdots w_k)$, where $t_s$ is the start time, $t_e$ is the end time, and $w_1 w_2 \cdots w_k$ is a sequence of words. PLSA is thus used as a feature extractor that could potentially capture "topic turns" in meetings.

We embedded PLSA into our model by treating the states of individual players as instances of PLSA topics (similar to [5]). Therefore, the PLSA model determines the emission probability in Eq. 5. We repeat the PLSA topic within the same utterance ($t_s \leq t \leq t_e$). The topic for the silence segments was set to 0 (Fig. 4). We can see that using audio-only features can be seen as a special case of using language features, by using only one topic in the PLSA model (i.e., all utterances belong to the same topic). We set 10 topics in PLSA ($N_S = 10$), and set $N_G = 5$ using simple reasonable a priori knowledge. EM iterations were stopped once the relative difference in the global log likelihood was less than 2%.

## 5.3 Results and Discussions

We compare our model with a method based on the speaking length (how much time each of the participants speaks). In this case, the influence value of a meeting participant is defined to be proportional to his speaking length: $P(Q = i) = L_i / \sum_{i=1}^{N} L_i$, where $L_i$ is the speaking length of participant $i$. As a second baseline model, we randomly generated 1000 combinations of influence values (under the constraint that the sum of the four values equals 1), and report the average performance.

The results are shown in Table 1 (left) and Fig. 5(e-h). We can see that the results of the three methods: model + language, model + audio, and speaking-length (Fig. 5 (e-g)) are significantly better than the result of randomization (Fig. 5 (h)). Using language features

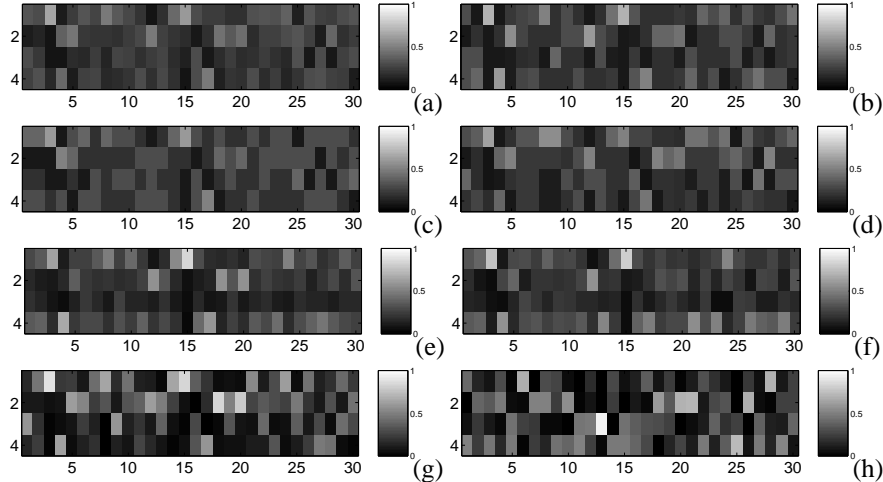

Figure 5: Influence values of the 4 participants (y-axis) in the 30 meetings (x-axis) (a) ground-truth (average of the three human annotations: $A_1, A_2, A_3$). (b) $A_1$ : human annotation 1 (c) $A_2$ : human annotation 2 (d) $A_3$ : human annotation 3 (e) our model + language (f) our model + audio (g) speaking-length (h) randomization.

Table 1: Results on meetings ("model" denotes the two-level influence model).

| Method | KL divergence |
|---|---|
| model + Language | 0.106 |
| model + Audio | 0.135 |
| Speaking length | 0.226 |
| Randomization | 0.863 |

| Human Annotation | KL divergence |
|---|---|
| $A_i$ vs. $A_j$ | 0.090 |
| $A_i$ vs. $\overline{A_i}$ | 0.053 |
| $A_i$ vs. GT | 0.037 |

with our model achieves the best performance. Our model (using either audio or language features) outperforms the speaking-length based method, which suggests that the learned influence distributions are in better accordance with the influence distributions from human judgements. As shown in Fig. 4, using audio features can be seen as a special case of using language features. We use language features to capture "topic turns" by factorizing the two states: "speaking, silence" into more states: "topic1, topic2, ..., silence". We can see that the result using language features is better than that using audio features. In other words, compared with "speaking turns", "topic turns" improves the performance of our model to learn the influence of participants in meetings.

It is interesting to look at the KL divergence between any pair of the three human annotations ($A_i$ vs. $A_j$), any one against the average of the others ($A_i$ vs. $\overline{A_i}$), and any one against the ground-truth ($A_i$ vs. GT). The average results are shown in Table 1 (right). We can see that the result of "$A_i$ vs. GT" is the best, which is reasonable since "GT" is the average of $A_1$, $A_2$, and $A_3$. Fig. 6(a) shows the histogram of KL divergence between any pair of human annotations for the 30 meetings. The histogram has a distribution of $\mu = 0.09, \sigma = 0.11$. We can see that the results of our model (language: 0.106, audio: 0.135) are very close to the mean ($\mu = 0.09$), which indicates that our model is comparable to human performance.

With our model, we can calculate the cumulative influence of each meeting participant over time. Fig. 6(b) shows such an example using the two-level influence model with audio features. We can see that the cumulative influence is related to the meeting agenda: The meeting starts with the monologue of person1 (monologue1). The influence of person1 is almost 1, while the influences of the other persons are nearly 0. When four participants are

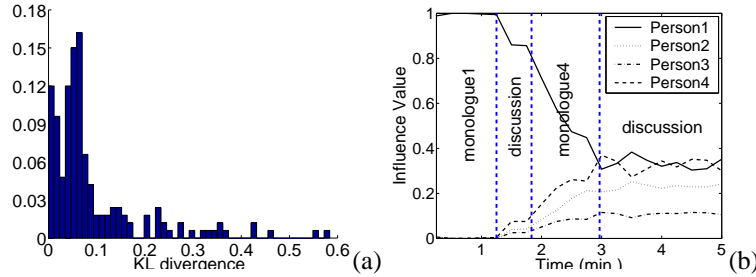

Figure 6: (a) Histogram of KL divergence between any pair of the human annotations ($A_i$ vs. $A_j$) for the 30 meetings. (b) The evolution of cumulative influence over time (5 minutes). The dotted vertical lines indicate the predefined meeting agenda.

involved in a discussion, the influence of person1 decreases, and the influences of the other three persons increase. The influence of person4 increases quickly during monologue4. The final influence of participants becomes stable in the second discussion.

## 6 Conclusions

We have presented a two-level influence model that learns the influence of all players within a team. The model has a two-level structure: individual-level and group-level. Individual level models actions of individual players and group-level models the group as a whole. Experiments on synthetic multi-player games and a multi-party meeting corpus showed the effectiveness of the proposed model. More generally, we anticipate that our approach to multi-level influence modeling may provide a means for analyzing a wide range of social dynamics to infer patterns of emergent group behaviors.

## Acknowledgements

This work was supported by the Swiss National Center of Competence in Research on Interactive Multimodal Information Management (IM2), and the EC project AMI (Augmented Multi-Party Interaction) (pub. AMI-124). We thank Florent Monay (IDIAP) and Jeff Bilmes (University of Washington) for sharing PLSA code and the GMTK. We also thank the annotators for their efforts.

## References

[1] C. Asavathiratham. The influence model: A tractable representation for the dynamics of networked markov chains. *Ph.D. dissertation, Dept. of EECS, MIT, Cambridge*, 2000.

[2] S. Basu, T. Choudhury, B. Clarkson, and A. Pentland. Learning human interactions with the influence model. *MIT Media Laboratory Technical Note No. 539*, 2001.

[3] J. Bilmes. Dynamic bayesian multinets. In *Uncertainty in Artificial Intelligence*, 2000.

[4] J. Bilmes and G. Zweig. The graphical models toolkit: An open source software system for speech and time series processing. *Proc. ICASSP*, vol. 4:3916–3919, 2002.

[5] D. Blei and P. Moreno. Topic segmentation with an aspect hidden markov model. *Proc. of ACM SIGIR conference on Research and development in information retrieval*, pages 343–348, 2001.

[6] T. Choudhury and S. Basu. Modeling conversational dynamics as a mixed memory markov process. *Proc. of Intl. Conference on Neural Information and Processing Systems (NIPS)*, 2004.

[7] J.A. Cohen. A coefficient of agreement for nominal scales. *Educ Psych Meas*, 20:37–46, 1960.

[8] S. L. Ellyson and J. F. Dovidio, editors. *Power, Dominance, and Nonverbal Behavior*. Springer-Verlag., 1985.

[9] T. Hofmann. Unsupervised learning by probabilistic latent semantic analysis. In *Machine Learning, 42:177–196*, 2001.

[10] A. Howard and T. Jebara. Dynamical systems trees. In *Uncertainty in Artificial Intelligence'01*.

[11] K. Kirchhoff, S. Parandekar, and J. Bilmes. Mixed-memory markov models for automatic language identification. *IEEE Int. Conf. on Acoustics, Speech, and Signal Processing*, 2000.

[12] I. McCowan, D. Gatica-Perez, S. Bengio, G. Lathoud, M. Barnard, and D. Zhang. Automatic analysis of multimodal group actions in meetings. In *IEEE Transactions on PAMI*, volume 27(3), 2005.

[13] N. Oliver, B. Rosario, and A. Pentland. Graphical models for recognizing human interactions. *Proc. of Intl. Conference on Neural Information and Processing Systems (NIPS)*, 1998.

[14] L. K. Saul and M. I. Jordan. Mixed memory markov models: Decomposing complex stochastic processes as mixtures of simpler ones. *Machine Learning*, 37(1):75–87, 1999.
